# The Diffusion Mediated Biochemical Signal Relay Channel

**Peter J. Thomas**,[*] **Donald J. Spencer**[†]
Computational Neurobiology Laboratory
(Terrence J. Sejnowski, Director)
Salk Institute for Biological Studies
La Jolla, CA 92037

**Sierra K. Hampton, Peter Park, Joseph P. Zurkus**
Department of Electrical and Computer Engineering
University of California San Diego
La Jolla, CA 92093

## Abstract

Biochemical signal-transduction networks are the biological information-processing systems by which individual cells, from neurons to amoebae, perceive and respond to their chemical environments. We introduce a simplified model of a single biochemical relay and analyse its capacity as a communications channel. A diffusible ligand is released by a sending cell and received by binding to a transmembrane receptor protein on a receiving cell. This receptor-ligand interaction creates a nonlinear communications channel with non-Gaussian noise. We model this channel numerically and study its response to input signals of different frequencies in order to estimate its channel capacity. Stochastic effects introduced in both the diffusion process and the receptor-ligand interaction give the channel low-pass characteristics. We estimate the channel capacity using a water-filling formula adapted from the additive white-noise Gaussian channel.

## 1 Introduction: The Diffusion-Limited Biochemical Signal-Relay Channel

The term *signal-transduction network* refers to the web of biochemical interactions by which single cells process sensory information about their environment. Just as neural networks underly the interaction of many multicellular organisms with their environments, these biochemical networks allow cells to perceive, evaluate and react to chemical stimuli [1]. Examples include chemical signaling across the synaptic cleft, calcium signaling within the postsynaptic dendritic spine, pathogen localization by the immune system,

---

[*]Corresponding author: pjthomas@salk.edu
[†]dspencer@salk.edu

growth-cone guidance during neuronal development, phototransduction in the retina, rhythmic chemotactic signaling in social amoebae, and many others. The introduction of quantitative measurements of the distribution and activation of chemical reactants within living cells [2] has prepared the way for detailed quantitative analysis of their properties, aided by numerical simulations. One of the key questions that can now be addressed is the fundamental limits to cell-to-cell communication using chemical signaling.

To communicate via chemical signaling cells must contend with the unreliability inherent in chemical diffusion and in the interactions of limited numbers of signaling molecules and receptors [3]. We study a simplified situation in which one cell secretes a signaling molecule, or *ligand*, which can be detected by a receptor on another cell. Limiting ourselves to one ligand-receptor interaction allows a treatment of this communications system using elementary concepts from information theory.

The information capacity of this fundamental signaling system is the maximum of the mutual information between the ensemble of input signals, the time-varying rate of ligand secretion $s(t)$, and the output signal $r(t)$, a piecewise continuous function taking the values one or zero as the receptor is bound to ligand or unbound. Using numerical simulation we can estimate the channel capacity via a standard "water-filling" information measure [4], as described below.

## 2 Methods: Numerical Simulation of the Biochemical Relay

We simulate a biochemical relay system as follows: in a two-dimensional rectangular volume $V$ measuring 5 micrometers by 10 micrometers, we locate two cells spaced 5 micrometers apart. Cell $A$ emits ligand molecules from location $x_s = [2.5\mu, 2.5\mu]$ with rate $s(t) \geq 0$; they diffuse with a given diffusion constant $D$ and decay at a rate $\alpha$. Both secretion and decay occur as random Poisson processes, and diffusion is realized as a discrete random walk with Gaussian-distributed displacements. The boundaries of $V$ are taken to be reflecting. We track the positions of each of $N$ particles $\{x_i, i = 1, \cdots, N\}$ at intervals of $\Delta t = 1$msec. The *local concentration* in a neighborhood of size $\sigma$ around a location $x$ is given by the convolution

$$\hat{c}(x,t) = \int_V \sum_{i=1}^N \delta(x' - x_i) g(x - x', \sigma) \, dx' \tag{1}$$

where $g(\cdot, \sigma)$ is a normalized Gaussian distribution in the plane, with mean 0 and variance $\sigma^2$. The motions of the individual particles cause $\hat{c}(x,t)$ to fluctuate about the mean concentration, causing the local concentration at cell B, $\hat{c}(x_r, t)$ to be a noisy, low-pass filtered version of the original signal $s(t)$ (see Figure 1).

Cell B, located at $x_r = [7.5\mu, 2.5\mu]$, registers the presence of ligand through binding and unbinding transitions, which form a two-state Markov process with time-varying transition rates. Given an unbound receptor, the binding transition happens at a rate that depends on the ligand concentration around the receptor: $k_+ \hat{c}(x_r, t)$. The size of the neighborhood $\sigma$ reflects the range of the receptor, with binding most likely in a small region close to $x_r$. Once the receptor is bound to a ligand molecule, no more binding events occur until the receptor releases the ligand. The receiver is insensitive to fluctuations in $\hat{c}(x_r, t)$ while it is in the bound state (see Figure 1). The unbinding transition occurs with a fixed rate $k_-$.

For concreteness, we take values for $D, \alpha, k_-, k_+$, and $\sigma$ appropriate for cyclic AMP signaling between *Dictyostelium* amoebae, a model organism for chemical communication: $D = 0.25\mu^2$msec$^{-1}$, $\alpha = 1$ sec$^{-1}$, $\sigma = 0.1\mu$, $k_- = 1$ sec$^{-1}$, $k_+ = \frac{1}{2\pi\sigma^2}$ sec$^{-1}$. $K_d = k_-/k_+$ is the dissociation constant, the concentration at which the receptor on average is bound half the time. For the chosen values of the reaction constants $k_\pm$, we have

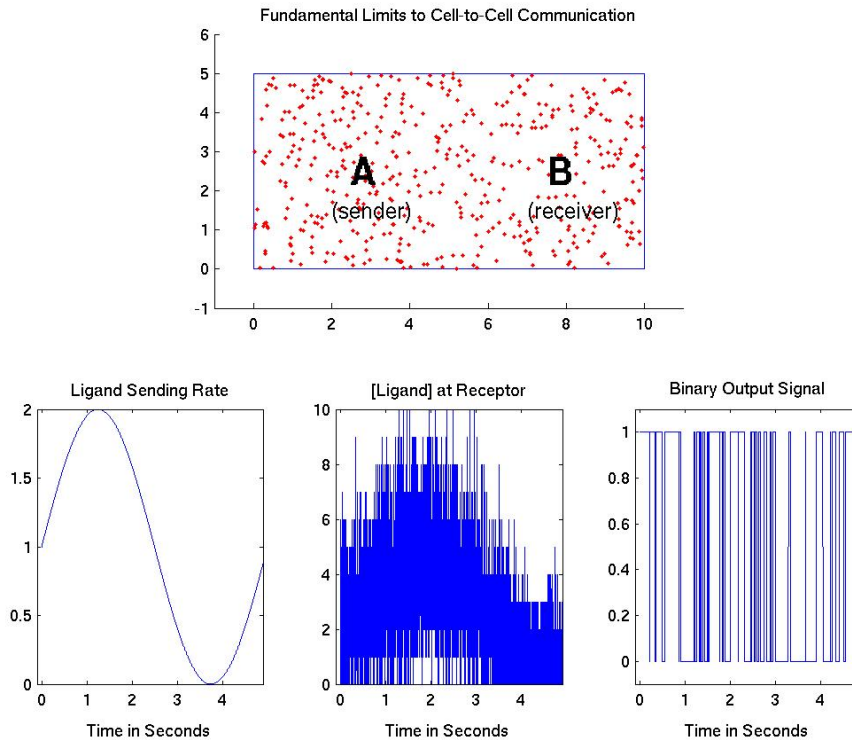

Figure 1: **Biochemical Signaling Simulation.**
**Top:** Cell **A** secretes a signaling molecule (red dots) with a time-varying rate $r(t)$. Molecules diffuse throughout the two-dimensional volume, leading to locally fluctuating concentrations that carry a corrupted version of the signal. Molecules within a neighborhood of cell **B** can bind to a receptor molecule, giving a received signal $s(t) \in \{0, 1\}$.
**Bottom Left:** Input signal. Mean instantaneous rate of molecule release (thousands of molecules per second). Molecule release is a Poisson process with time-varying rate.
**Bottom Center:** Local concentration fluctuations, as seen by cell B, indicated by the number of molecules within 0.2 microns of the receptor. The receptor is sensitive to fluctuations in local concentrations only while it is unbound. While the receptor is bound, it does not register changes in the local concentration (indicated by constant plateaus corresponding to intervals when $r(t) = 1$ in bottom right panel.
**Bottom Right:** Output signal $r(t)$. At each moment the receptor is either bound (1) or unbound (0). The receiver output is a piecewise constant function with a finite number of transitions.

$K_d \approx 15.9 \frac{\text{molecules}}{\mu^2} \approx 26.4 \text{nMol}$, comparable to the most sensitive values reported for the cyclic AMP receptor [2]. At this concentration the volume $V = 50\mu^2$ contains about 800 signaling molecules, assuming a nominal depth of $1\mu$.

## 3   Results: Estimating Information Capacity via Frequency Response

Communications channels mediated by diffusion and ligand receptor interaction are non-linear with non-Gaussian noise. The expected value of the output signal, $0 \leq E[r] < 1$, is a sigmoidal function of the log concentration for a constant concentration $c$:

$$E[r] = \frac{c}{c + K_d} = \frac{1}{1 + e^{-(y-y_0)}} \tag{2}$$

where $y = \ln(c), y_0 = \ln(K_d)$. The mean response saturates for high concentrations, $c \gg K_d$, and the noise statistics become pronouncedly Poissonian (rather than Gaussian) for low concentrations.

Several different kinds of stimuli can be used to characterize such a channel. The steady-state response to constant input reflects the static (equilibrium) transfer function. Concentrations ranging from $100K_d$ to $0.01K_d$ occupy 98% of the steady-state operating range, $0.99 > E[r] > 0.01$ [5]. For a finite observation time $T$ the actual fraction of time spent bound, $\bar{r}_T$, is distributed about $E[r]$ with a variance that depends on $T$. The biochemical relay may be used as a binary symmetric channel randomly selecting a 'high' or 'low' secretion rate, and 'decoding' by setting a suitable threshold for $\bar{r}_T$. As $T$ increases, the variance of $\bar{r}_T$ and the probability of error decrease.

The binary symmetric channel makes only crude use of this signaling mechanism. Other possible communication schemes include sending all-or-none bursts of signaling molecule, as in synaptic transmission, or detecting discrete stepped responses. Here we use the *frequency response* of the channel as a way of estimating the information capacity of the biochemical channel.

For an idealized linear channel with additive white Gaussian noise (AWNG channel) the channel capacity under a mean input power constraint $P$ is given by the so-called "water-filling formula" [4],

$$C = \frac{1}{2} \int_{\omega=\omega_{\min}}^{\omega_{\max}} \log_2 \left( 1 + \frac{(\nu - N(\omega))^+}{N(\omega)} \right) d\omega \tag{3}$$

given the constraining condition

$$\int_{\omega=\omega_{\min}}^{\omega_{\max}} (\nu - N(\omega))^+ d\omega \leq P \tag{4}$$

where the constant $\nu$ is the sum of the noise and the signal power in the usable frequency range, $N(\omega)$ is the power of the additive noise at frequency $\omega$ and $(X)^+$ indicates the positive part of $X$. The formula applies when each frequency band $(\omega, \omega + d\omega)$ is subject to noise of power $N(\omega)$ independently of all other frequency bands, and reflects the optimal allocation of signal power $S(\omega) = (\nu - N(\omega))^+$, with greater signal power invested in frequencies at which the noise power is smallest. The capacity $C$ is in bits/second.

For an input signal of finite duration $T = 100$ sec, we can independently specify the amplitudes and phases of its frequency components at $\omega = [0.01 \text{ Hz}, 0.02 \text{ Hz}, \cdots, 500 \text{ Hz}]$, where 500 Hz is the Nyquist frequency given a 1 msec simulation timestep. Because the population of secreted signaling molecules decays exponentially with a time constant of $1/\alpha = 1$ sec, the concentration signal is unable to pass frequencies $\omega \geq 1$ Hz (see Figure 2) providing a natural high-frequency cutoff. For the AWGN channel the input and

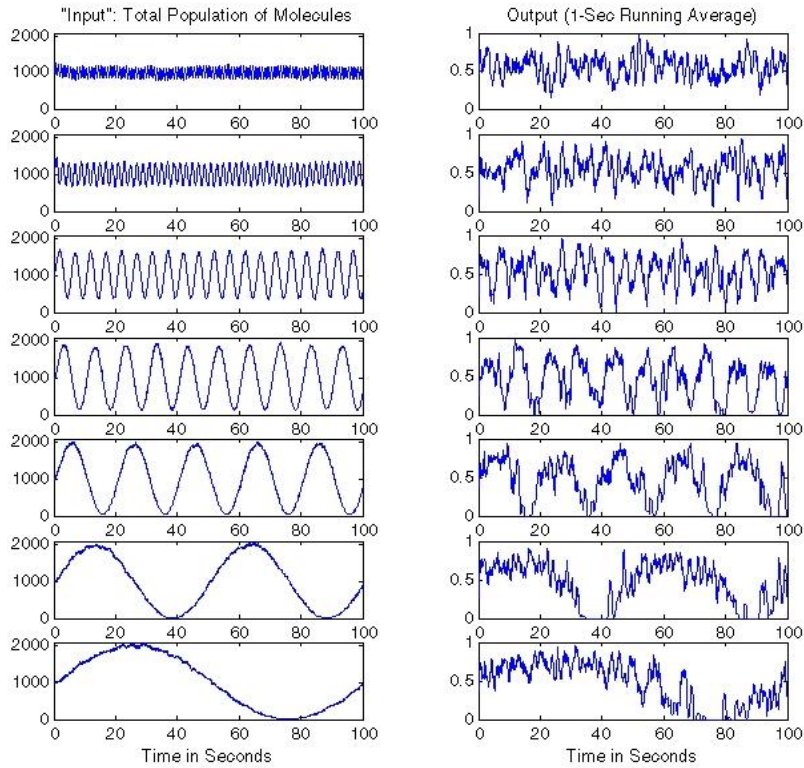

Figure 2: **Frequency Response of Biochemical Relay Channel.** The sending cell secreted signaling molecules at a mean rate of $1000 + 1000\sin(2\pi\omega t)$ molecules per second. From top to bottom, the input frequencies were 1.0, 0.5, 0.2, 0.1, 0.05, 0.02 and 0.01 Hz. The total signal duration was $T = 100$ seconds.

**Left Column:** Total number of molecules in the volume. Attenuation of the original signal results from exponential decay of the signaling molecule population.

**Right Column:** A one-second moving average of the output signal $r(t)$, which takes the value one when the receptor molecule is bound to ligand, and zero when the receptor is unbound.

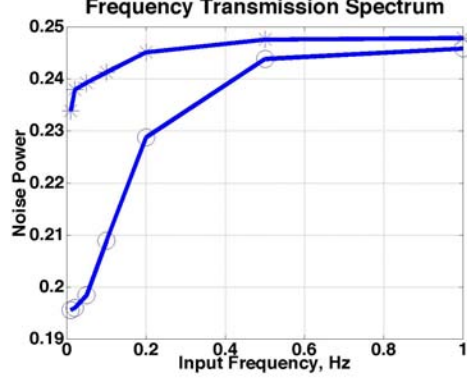

Figure 3: **Frequency Transmission Spectrum** Noise power $N(\omega)$, calculated as the total power in $r(t) - \bar{r}$ in all frequency components save the input frequency $\omega$. Frequencies were binned in intervals of 0.01 Hz = $1/T$. The maximum possible power in $r(t)$ over all frequencies is 0.25; the power successfully transmitted by the channel is given by $0.25/N(\omega)$. The lower curve is $N(\omega)$ for input signals of the form $s(t) = 1000 + 1000\sin 2\pi\omega t$, which uses the full dynamic range of the receptor. Decreasing the dynamic range used reduces the amount of power transmitted at the sending frequency: the upper curve is $N(\omega)$ for signals of the form $s(t) = 1000 + 500\sin 2\pi\omega t$.

output signals share the same units (e.g. rms voltage); for the biological relay the input $s(t)$ is in molecules/second while the output $r(t)$ is a function with binary range $\{r = 0, r = 1\}$. The maximum of the mean output power for a binary function $r(t)$ is $\left(\frac{1}{T}\int_{t=0}^{T}|r(t) - \bar{r}|\, dt\right)^2 \leq \frac{1}{4}$. This total possible output power will be distributed between different frequencies depending on the frequency of the input. We wish to estimate the channel capacity by comparing the portion of the output power present in the sending frequency $\omega$ to the limiting output power 0.25. Therefore we set the total output power constant to $\nu = 0.25$. Given a pure sinusoidal input signal $s(t) = a_0 + a_1\sin(2\pi\omega t)$, we consider the power in the output spectrum at $\omega$ Hz to be the residual power from the input and the rest of the power in the spectrum of $r(t)$ to be analogous to the additive noise power spectrum $N(\omega)$ in the AWNG channel. We calculate $N(\omega)$ to be the total power of $r(t) - \bar{r}$ in all frequency bands except $\omega$. For signals of length $T = 100$ sec, the possible frequencies are discretized at intervals $\Delta\omega = 0.01$ Hz. Because the noise power $N(\omega) \leq 0.25$, the water-filling formula (3) for the capacity reduces to

$$C_{\text{est}} = \frac{1}{2}\int_{0.01\text{Hz}}^{1\text{Hz}} \log_2\left(\frac{0.25}{N(\omega)}\right)\, d\omega. \tag{5}$$

As mentioned above frequencies $\omega \geq 1$ Hz do not transmit any information about the signal (see Figure 2) and do not contribute to the capacity. We approximate this integral using linear interpolation of $\log_2(N(\omega))$ between the measured values at $\omega = [0.01, 0.02, 0.05, 0.1, 0.2, 0.5, 1.0]$ Hz. (See Figure 3.) This procedure gives an estimate of the channel capacity, $C_{\text{est}} = 0.087$ bits/second.

## 4 Discussion & Conclusions

Diffusion and the Markov switching between bound and unbound states create a low-pass filter that removes high-frequency information in the biochemical relay channel. A general

Poisson-type communications channel, such as commonly encountered in optical communications engineering, can achieve an arbitrarily large capacity by transmitting high frequencies and high amplitudes, unless bounded by a max or mean amplitude constraint [6]. In the biochemical channel, the effective input amplitude is naturally constrained by the saturation of the receptor at concentrations above the $K_d$. And the high frequency transmission is limited by the inherent dynamics of the Markov process. Therefore this channel has a finite capacity.

The channel capacity estimate we derived, $C_{\text{est}} = 0.087$ bits/second, seems quite low compared to signaling rates in the nervous system, requiring long signaling times to transfer information successfully. However temporal dynamics in cellular systems can be quite deliberate; cell-cell communication in the social amoeba *Dictyostelium*, for example, is achieved by means of a carrier wave with a period of seven minutes. In addition, cells typically possess thousands of copies of the receptors for important signaling molecules, allowing for more complex detection schemes than those investigated here.

Our simplified treatment suggests several avenues for further work. For example, signal transducing receptors often form Markov chains with more complicated dynamics reflecting many more than two states [7]. Also, the nonlinear nature of the channel is probably not well served by our additive noise approximation, and might be better suited to a treatment via multiplicative noise [8].

Whether cells engage in complicated temporal coding/decoding schemes, as has been proposed for neural information processing, or whether instead they achieve efficient communication by evolutionary matching of the noise characteristics of sender and receiver, remain to be investigated. We note that the dependence of the channel capacity $C$ on such parameters as the system geometry, the diffusion and decay constants, the binding constants and the range of the receptor may shed light on evolutionary mechanisms and constraints on communication within cellular biological systems.

## Acknowledgments

This work would not have been possible without the generous support of the Howard Hughes Medical Institute and the resources of the Computational Neurobiology Laboratory, Terrence J. Sejnowski, Director.

## References

[1] Rappel, W.M., Thomas, P.J., Levine, H. & Loomis, W.F. (2002) Establishing Direction during Chemotaxis in Eukaryotic Cells. *Biophysical Journal* **83**:1361-1367.

[2] Ueda, M., Sako, Y., Tanaka, T., Devreotes, P. & Yanagida, T. (2001) Single Molecule Analysis of Chemotactic Signaling in Dictyostelium Cells. *Science* **294**:864-867.

[3] Detwiler, P.B., Ramanathan, S., Sengupta, A. & Shraiman, B.I. (2000) Engineering Aspects of Enzymatic Signal Transduction: Photoreceptors in the Retina. *Biophysical Journal* **79**:2801-2817.

[4] Cover, T.M. & Thomas, J.A. (1991) *Elements of Information Theory*, New York: Wiley.

[5] Getz, W.M. & Lansky, P. (2001) Receptor Dissociation Constants and the Information Entropy of Membranes Coding Ligand Concentration. *Chem. Senses* **26**:95-104.

[6] Frey, R.M. (1991) Information Capacity of the Poisson Channel. *IEEE Transactions on Information Theory* **37**(2):244-256.

[7] Uteshev, V.V. & Pennefather, P.S. (1997) Analytical Description of the Activation of Multi-State Receptors by Continuous Neurotransmitter Signals at Brain Synapses. *Biophysical Journal* **72**:1127-1134.

[8] Mitra, P.P. & Stark, J.B. (2001) Nonlinear limits to the information capacity of optical fibre

communications. *Nature* **411**:1027-1030.